# Balancing between bagging and bumping

**Tom Heskes**
RWCP Novel Functions SNN Laboratory,* University of Nijmegen
Geert Grooteplein 21, 6525 EZ Nijmegen, The Netherlands
tom@mbfys.kun.nl

## Abstract

We compare different methods to combine predictions from neural networks trained on different bootstrap samples of a regression problem. One of these methods, introduced in [6] and which we here call balancing, is based on the analysis of the ensemble generalization error into an ambiguity term and a term incorporating generalization performances of individual networks. We show how to estimate these individual errors from the residuals on validation patterns. Weighting factors for the different networks follow from a quadratic programming problem. On a real-world problem concerning the prediction of sales figures and on the well-known Boston housing data set, balancing clearly outperforms other recently proposed alternatives as bagging [1] and bumping [8].

## 1 EARLY STOPPING AND BOOTSTRAPPING

Stopped training is a popular strategy to prevent overfitting in neural networks. The complete data set is split up into a training and a validation set. Through learning the weights are adapted in order to minimize the error on the training data. Training is stopped when the error on the validation data starts increasing. The final network depends on the accidental subdivision in training and validation set, and often also on the, usually random, initial weight configuration and chosen minimization procedure. In other words, early stopped neural networks are highly unstable: small changes in the data or different initial conditions can produce large changes in the estimate. As argued in [1, 8], with unstable estimators it is advisable to resample, i.e., to apply the same procedure several times using different subdivisions in training and validation set and perhaps starting from different initial

configurations. In the neural network literature resampling is often referred to as training ensembles of neural networks [3, 6]. In this paper, we will discuss methods for combining the outputs of networks obtained through such a repetitive procedure.

First, however, we have to choose how to generate the subdivisions in training and validation sets. Options are, among others, k-fold cross-validation, subsampling and bootstrapping. In this paper we will consider bootstrapping [2] which is based on the idea that the available data set is nothing but a particular realization of some probability distribution. In principle, one would like to do inference on this "true" yet unknown probability distribution. A natural thing to do is then to define an empirical distribution. With so-called naive bootstrapping the empirical distribution is a sum of delta peaks on the available data points, each with probability content $1/p_{\text{data}}$ with $p_{\text{data}}$ the number of patterns. A bootstrap sample is a collection of $p_{\text{data}}$ patterns drawn with replacement from this empirical probability distribution. Some of the data points will occur once, some twice and some even more than twice in this bootstrap sample. The bootstrap sample is taken to be the training set, all patterns that do not occur in a particular bootstrap sample constitute the validation set. For large $p_{\text{data}}$, the probability that a pattern becomes part of the validation set is $(1 - 1/p_{\text{data}})^{p_{\text{data}}} \approx 1/e \approx 0.368$. An advantage of bootstrapping over other resampling techniques is that most statistical theory on resampling is nowadays based on the bootstrap.

Using naive bootstrapping we generate $n_{\text{run}}$ training and validation sets out of our complete data set of $p_{\text{data}}$ input-output combinations $\{\vec{x}^{\mu}, t^{\mu}\}$. In this paper we will restrict ourselves to regression problems with, for notational convenience, just one output variable. We keep track of a matrix with components $q_i^{\mu}$ indicating whether pattern $\mu$ is part of the validation set for run $i$ ($q_i^{\mu} = 1$) or of the training set ($q_i^{\mu} = 0$). On each subdivision we train and stop a neural network with one layer of $n_{\text{hidden}}$ hidden units. The output $o_i^{\mu}$ of network $i$ with weight vector $\mathbf{w}(i)$ on input $\vec{x}^{\mu}$ reads

$$o_i^{\mu} = \sum_{j=1}^{n_{\text{hidden}}} w_j(i) \tanh \left[ \sum_{k=0}^{n_{\text{input}}} w_{(k+1)n_{\text{hidden}}+j}(i) x_k^{\mu} \right] + w_0(i) ,$$

where we use the definition $x_0^{\mu} \equiv 1$. The validation error for run $i$ can be written

$$E_{\text{validation}}(i) \equiv \frac{1}{p_i} \sum_{\mu=1}^{p_{\text{data}}} q_i^{\mu} r_i^{\mu} ,$$

with $p_i \equiv \sum_{\mu} q_i^{\mu} \approx 0.368 \, p_{\text{data}}$, the number of validation patterns in run $i$, and $r_i^{\mu} \equiv (o_i^{\mu} - t^{\mu})^2/2$, the error of network $i$ on pattern $\mu$.

After training we are left with $n_{\text{run}}$ networks, with, in practice, quite different performances on the complete data set. How should we combine all these outputs to get the best possible performance on new data?

## 2 COMBINING ESTIMATORS

Several methods have been proposed to combine estimators (see e.g. [5] for a review). In this paper we will only consider estimators with the same architecture

but trained and stopped on different subdivisions of the data in training and validation sets. Recently, two such methods have been suggested for bootstrapped estimators: bagging [1], an acronym for bootstrap aggregating, and bumping [8], meaning bootstrap umbrella of model parameters. With bagging, the prediction on a newly arriving input vector is the average over all network predictions. Bagging completely disregards the performance of the individual networks on the data used for training and stopping. Bumping, on the other hand, throws away all networks except the one with the lowest error on the complete data set[1]. In the following we will describe an intermediate form due to [6], which we here call balancing. A theoretical analysis of the implications of this idea can be found in [7].

Suppose that after training we receive a new set of $p_{\text{test}}$ test patterns for which we do not know the true targets $\tilde{t}^\nu$, but can calculate the network output $\tilde{o}_i$ for each network $i$. We give each network a weighting factor $\alpha_i$ and define the prediction of all networks on pattern $\nu$ as the weighted average

$$\tilde{m}^\nu \equiv \sum_{i=1}^{n_{\text{run}}} \alpha_i \tilde{o}_i^\nu \, .$$

The goal is to find the weighting factors $\alpha_i$, subject to the constraints

$$\sum_{i=1}^{n_{\text{run}}} \alpha_i = 1 \quad \text{and} \quad \alpha_i \geq 0 \ \forall_i \, , \tag{1}$$

yielding the smallest possible generalization error

$$E_{\text{test}} \equiv \frac{1}{p_{\text{test}}} \sum_{\nu=1}^{p_{\text{test}}} (\tilde{m}^\nu - \tilde{t}^\nu)^2 \, .$$

The problem, of course, is our ignorance about the targets $\tilde{t}^\nu$. Bagging simply takes $\alpha_i = 1/n_{\text{run}}$ for all networks, whereas bumping implies $\alpha_i = \delta_{i\kappa}$ with

$$\kappa = \underset{i}{\text{argmin}} \frac{1}{p_{\text{data}}} \sum_{\mu=1}^{p_{\text{data}}} (o_i^\mu - t^\mu)^2 \, .$$

As in [6, 7] we write the generalization error in the form

$$
\begin{aligned}
E_{\text{test}} &= \frac{1}{p_{\text{test}}} \sum_\nu \sum_{i,j} \alpha_i \alpha_j (\tilde{o}_i^\nu - \tilde{t}^\nu)(\tilde{o}_j^\nu - \tilde{t}^\nu) \\
&= \frac{1}{2p_{\text{test}}} \sum_\nu \sum_{i,j} \alpha_i \alpha_j \left[ (\tilde{o}_i^\nu - \tilde{t}^\nu)^2 + (\tilde{o}_j^\nu - \tilde{t}^\nu)^2 - (\tilde{o}_i^\nu - \tilde{o}_j^\nu)^2 \right] \\
&= \sum_{i,j} \alpha_i \alpha_j \left[ E_{\text{test}}(i) + E_{\text{test}}(j) - \frac{1}{2p_{\text{test}}} \sum_\nu (\tilde{o}_i^\nu - \tilde{o}_j^\nu)^2 \right] \, . \tag{2}
\end{aligned}
$$

The last term depends only on the network outputs and can thus be calculated. This "ambiguity" term favors networks with conflicting outputs. The first part,

containing the generalization errors $E_{\text{test}}(i)$ for individual networks, depends on the targets $\tilde{t}^\nu$ and is thus unknown. It favors networks that by themselves already have a low generalization error. In the next section we will find reasonable estimates for these generalization errors based on the network performances on validation data. Once we have obtained these estimates, finding the optimal weighting factors $\alpha_i$ under the constraints (1) is a straightforward quadratic programming problem.

## 3 ESTIMATING THE GENERALIZATION ERROR

At first sight, a good estimate for the generalization error of network $i$ could be the performance on the validation data not included during training. However, the validation error $E_{\text{validation}}(i)$ strongly depends on the accidental subdivision in training and validation set. For example, if there are a few outliers which, by pure coincidence, are part of the validation set, the validation error will be relatively large and the training error relatively small. To correct for this bias as a result of the random subdivision, we introduce the "expected" validation error for run $i$. First we define $n^\mu$ as the number of runs in which pattern $\mu$ is part of the validation set and $E^\mu_{\text{validation}}$ as the error averaged over these runs:

$$n^\mu \equiv \sum_{i=1}^{n_{\text{run}}} q_i^\mu \quad \text{and} \quad E^\mu_{\text{validation}} \equiv \frac{1}{n^\mu} \sum_{i=1}^{n_{\text{run}}} q_i^\mu r_i^\mu \,,$$

The expected validation error then follows from

$$\hat{E}_{\text{validation}}(i) \equiv \frac{1}{p_i} \sum_{\mu=1}^{p_{\text{data}}} q_i^\mu E^\mu_{\text{validation}} \,.$$

The ratio between the observed and the expected validation error indicates whether the validation error for network $i$ is relatively high or low. Our estimate for the generalization error of network $i$ is this ratio multiplied by an overall scaling factor being the estimated average generalization error:

$$E_{\text{test}}(i) \approx \frac{E_{\text{validation}}(i)}{\hat{E}_{\text{validation}}(i)} \frac{1}{p_{\text{data}}} \sum_{\mu=1}^{p_{\text{data}}} E^\mu_{\text{validation}} \,.$$

Note that we implicitly make the assumption that the bias introduced by stopping at the minimal error on the validation patterns is negligible, i.e., that the validation patterns used for stopping a network can be considered as new to this network as the completely independent test patterns.

## 4 SIMULATIONS

We compare the following methods for combining neural network outputs.

**Individual:** the average individual generalization error, i.e., the generalization error we will get on average when we decide to perform only one run. It serves as a reference with which the other methods will be compared.

**Bumping:** the generalization of the network with the lowest error on the data available for training and stopping.

| | bumping | bagging | ambiguity | balancing | unfair bumping | unfair balancing |
|---|---|---|---|---|---|---|
| store 1 | 4 % | 9 % | 10 % | 17 % | 17 % | 24 % |
| store 2 | 5 % | 15 % | 22 % | 23 % | 23 % | 34 % |
| store 3 | -7 % | 11 % | 18 % | 25 % | 25 % | 36 % |
| store 4 | 6 % | 11 % | 17 % | 26 % | 26 % | 31 % |
| store 5 | 6 % | 10 % | 22 % | 19 % | 22 % | 26 % |
| store 6 | 1 % | 8 % | 14 % | 19 % | 16 % | 26 % |
| mean | 3 % | 11 % | 17 % | 22 % | 22 % | 30 % |

Table 1: Decrease in generalization error relative to the average individual generalization error as a result of several methods for combining neural networks trained to predict the sales figures for several stores.

**Bagging:** the generalization error when we take the average of all $n_{\text{run}}$ network outputs as our prediction.

**Ambiguity:** the generalization error when the weighting factors are chosen to maximize the ambiguity, i.e., taking identical estimates for the individual generalization errors of all networks in expression (2).

**Balancing:** the generalization error when the weighting factors are chosen to minimize our estimate of the generalization error.

**Unfair bumping:** the smallest generalization error for an individual error, i.e., the result of bumping if we had indeed chosen the network with the smallest generalization error.

**Unfair balancing:** the lowest possible generalization error that we could obtain if we had perfect estimates of the individual generalization errors.

The last two methods, unfair bumping and unfair balancing, only serve as some kind of reference and can never be used in practice.

We applied these methods on a real-world problem concerning the prediction of sales figures for several department stores in the Netherlands. For each store, 100 networks with 4 hidden units were trained and stopped on bootstrap samples of about 500 patterns. The test set, on which the performances of the various methods for combination were measured, consists of about 100 patterns. Inputs include weather conditions, day of the week, previous sales figures, and season. The results are summarized in Table 1, where we give the decrease in the generalization error relative to the average individual generalization error.

As can be seen in Table 1, bumping hardly improves the performance. The reason is that the error on the data used for training and stopping is a lousy predictor of the generalization error, since some amount of overfitting is inevitable. The generalization performance obtained through bagging, i.e., first averaging over all outputs, can be proven to be always better than the average individual generalization error.

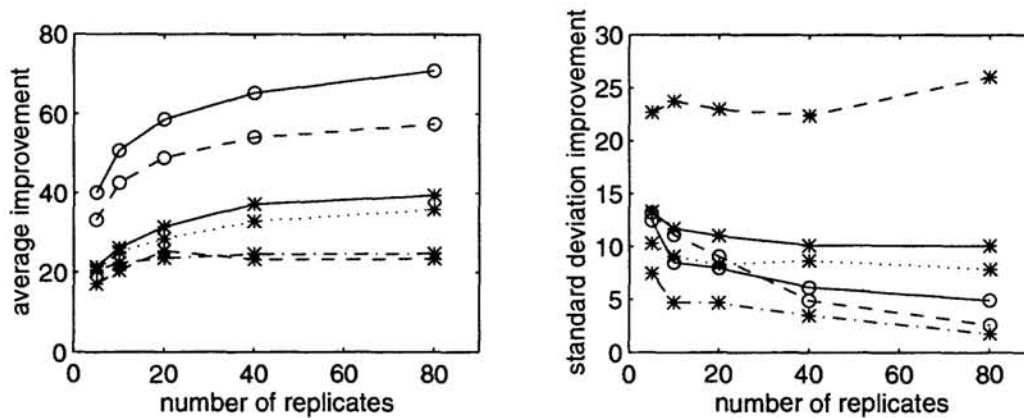

Figure 1: Decrease of generalization error relative to the average individual generalization error as a function of the number of bootstrap replicates for different combination methods: bagging (dashdot, star), ambiguity (dotted, star), bumping (dashed, star), balancing (solid, star), unfair bumping (dashed, circle), unfair balancing (solid, circle). Shown are the mean (left) and the standard deviation (right) of the decrease in percentages. Networks are trained and tested on the Boston housing database.

On these data bagging is definitely better than bumping, but also worse than maximizing the ambiguity. In all cases, except for store 5 where maximization of the ambiguity is slightly better, balancing is a clear winner among the "fair" methods. The last column in Table 1 shows how much better we can get if we could find more accurate estimates for the generalization errors of individual networks.

The method of balancing discards most of the networks, i.e., the solution to the quadratic programming problem (2) under constraints (1) yields just a few weighting factors different from zero (on average about 8 for this set of simulations). Balancing is thus indeed a compromise between bagging, taking all networks into account, and bumping, keeping just one network.

We also compared these methods on the well-known Boston housing data set concerning the median housing price in several tracts based on 13 mainly socio-economic predictor variables (see e.g. [1] for more information). We left out 50 of the 506 available cases for assessment of the generalization performance. All other 456 cases were used for training and stopping neural networks with 4 hidden units. The average individual mean squared error over all 300 bootstrap runs is 16.2, which is comparable to the mean squared error reported in [1]. To study how the performance depends on the number of bootstrap replicates, we randomly drew sets of $n = 5, 10, 20, 40$ and $80$ bootstrap replicates out of our ensemble of 300 replicates and applied the combination methods on these sets. For each $n$ we did this 48 times. Figure 1 shows the mean decrease in the generalization error relative to the average individual generalization error and its standard deviation.

Again, balancing comes out best, especially for a larger number of bootstrap replicates. It seems that beyond say 20 replicates both bumping and bagging are hardly helped by more runs, whereas both maximization of the ambiguity and balancing still increase their performance. Bagging, fully taking into account all network pre-

dictions, yields the smallest variation, bumping, keeping just one of them, by far the largest. Balancing and maximization of the ambiguity combine several predictions and thus yield a variation that is somewhere in between.

## 5   CONCLUSION AND DISCUSSION

Balancing, a compromise between bagging and bumping, is an attempt to arrive at better performances on regression problems. The crux in all this is to obtain reasonable estimates for the quality of the different networks and to incorporate these estimates in the calculation of the proper weighting factors (see [5, 9] for similar ideas and related work in the context of stacked generalization).

Obtaining several estimators is computationally expensive. However, the notorious instability of feedforward neural networks hardly leaves us a choice. Furthermore, an ensemble of bootstrapped neural networks can also be used to deduce (approximate) confidence and prediction intervals (see e.g. [4]), to estimate the relevance of input fields and so on. It has also been argued that combination of several estimators destroys the structure that may be present in a single estimator [8]. Having hardly any interpretable structure, neural networks do not seem to have a lot they can lose. It is a challenge to show that an ensemble of neural networks does not only give more accurate predictions, but also reveals more information than a single network.

## Footnotes

* RWCP: Real World Computing Partnership; SNN: Foundation for Neural Networks.

[1]The idea behind bumping is more general and involved than discussed here. The interested reader is referred to [8]. In this paper we will only consider its naive version.

## References

[1] L. Breiman. Bagging predictors. *Machine Learning*, 24:123–140, 1996.

[2] B. Efron and R. Tibshirani. *An Introduction to the Bootstrap*. Chapman & Hall, London, 1993.

[3] L. Hansen and P. Salomon. Neural network ensembles. *IEEE Transactions on Pattern Analysis and Machine Intelligence*, 12:993–1001, 1990.

[4] T. Heskes. Practical confidence and prediction intervals. *These proceedings*, 1997.

[5] R. Jacobs. Methods for combining experts' probability assessments. *Neural Computation*, 7:867–888, 1995.

[6] A. Krogh and J. Vedelsby. Neural network ensembles, cross validation, and active learning. In G. Tesauro, D. Touretzky, and T. Leen, editors, *Advances in Neural Information Processing Systems 7*, pages 231–238, Cambridge, 1995. MIT Press.

[7] P. Sollich and A. Krogh. Learning with ensembles: How over-fitting can be useful. In D. Touretzky, M. Mozer, and M. Hasselmo, editors, *Advances in Neural Information Processing Systems 8*, pages 190–196, San Mateo, 1996. Morgan Kaufmann.

[8] R. Tibshirani and K. Knight. Model search and inference by bootstrap "bumping". Technical report, University of Toronto, 1995.

[9] D. Wolpert and W. Macready. Combining stacking with bagging to improve a learning algorithm. Technical report, Santa Fe Institute, Santa Fe, 1996.
